# Infinite Mixtures of Gaussian Process Experts

**Carl Edward Rasmussen** and **Zoubin Ghahramani**
Gatsby Computational Neuroscience Unit
University College London
17 Queen Square, London WC1N 3AR, England
edward,zoubin@gatsby.ucl.ac.uk
http://www.gatsby.ucl.ac.uk

## Abstract

We present an extension to the Mixture of Experts (ME) model, where
the individual experts are Gaussian Process (GP) regression models. Us-
ing an input-dependent adaptation of the Dirichlet Process, we imple-
ment a gating network for an infinite number of Experts. Inference in this
model may be done efficiently using a Markov Chain relying on Gibbs
sampling. The model allows the effective covariance function to vary
with the inputs, and may handle large datasets – thus potentially over-
coming two of the biggest hurdles with GP models. Simulations show
the viability of this approach.

## 1 Introduction

Gaussian Processes [Williams & Rasmussen, 1996] have proven to be a powerful tool for
regression. They combine the flexibility of being able to model arbitrary smooth functions
if given enough data, with the simplicity of a Bayesian specification that only requires in-
ference over a small number of readily interpretable hyperparameters – such as the length
scales by which the function varies along different dimensions, the contributions of signal
and noise to the variance in the data, etc. However, GPs suffer from two important limita-
tions. First, because inference requires inversion of an $n \times n$ covariance matrix where $n$ is
the number of training data points, they are computationally impractical for large datasets.
Second, the covariance function is commonly assumed to be stationary, limiting the mod-
eling flexibility. For example, if the noise variance is different in different parts of the input
space, or if the function has a discontinuity, a stationary covariance function will not be
adequate. Goldberg et al [1998] discussed the case of input dependent noise variance.

Several recent attempts have been aimed at approximate inference in GP models [Williams
& Seeger 2001, Smola & Bartlett 2001]. These methods are based on selecting a projection
of the covariance matrix onto a smaller subspace (e.g. a subset of the data points) reducing
the overall computational complexity. There have also been attempts at deriving more
complex covariance functions [Gibbs 1997] although it can be difficult to decide a priori
on a covariance function of sufficient complexity which guarantees positive definiteness.

In this paper we will simultaneously address both the problem of computational complexity
and the deficiencies in covariance functions using a divide and conquer strategy inspired
by the Mixture of Experts (ME) architecture [Jacobs et al, 1991]. In this model the input

space is (probabilistically) divided by a *gating network* into regions within which specific separate experts make predictions. Using GP models as experts we gain the double advantage that computation for each expert is cubic only in the number of data point in its region, rather than in the entire dataset, and that each GP-expert may learn different characteristics of the function (such as lengths scales, noise variances, etc). Of course, as in the ME, the learning of the experts and the gating network are intimately coupled.

Unfortunately, it may be (practically and statistically) difficult to infer the appropriate number of experts for a particular dataset. In the current paper we sidestep this difficult problem by using an infinite number of experts and employing a gating network related to the Dirichlet Process, to specify a spatially varying Dirichlet Process. An infinite number of experts may also in many cases be more faithful to our prior expectations about complex real-word datasets. Integrating over the posterior distribution for the parameters is carried out using a Markov Chain Monte Carlo approach.

Tresp [2001] presented an alternative approach to mixtures of GPs. In his approach both the $M$ experts and the gating network were implemented with GPs; the gating network being a softmax of $M$ GPs. Our new model avoids several limitations of the previous approach, which are covered in depth in the discussion.

## 2  Infinite GP mixtures

The traditional ME likelihood does not apply when the experts are non-parametric. This is because in a normal ME model the data is assumed to be iid given the model parameters:

$$p(\mathbf{y}|\mathbf{x},\theta) = \prod_i \sum_j p(y_i|c_i = j, x_i, \theta_j)p(c_i = j|x_i, \phi),$$

where $\mathbf{x}$ and $\mathbf{y}$ are inputs and outputs (boldface denotes vectors), $\theta_j$ are the parameters of expert $j$, $\phi$ are the parameters of the gating network and $c_i$ are the discrete indicator variables assigning data points to experts.

This iid assumption is contrary to GP models which solely model the dependencies in the joint distribution (given the hyperparameters). There is a joint distribution corresponding to every possible assignment of data points to experts; therefore the likelihood is a sum over (exponentially many) assignments:

$$\begin{aligned}
p(\mathbf{y}|\mathbf{x},\theta) &= \sum_{\mathbf{c}} p(\mathbf{y}|\mathbf{c},\mathbf{x},\theta)p(\mathbf{c}|\mathbf{x},\phi) \\
&= \sum_{\mathbf{c}} \Big[ \prod_j p(\{y_i : c_i = j\}|\{x_i : c_i = j\},\theta_j)\Big]p(\mathbf{c}|\mathbf{x},\phi).
\end{aligned} \tag{1}$$

Given the configuration $\mathbf{c} = (c_1,\ldots,c_n)$, the distribution factors into the product, over experts, of the joint Gaussian distribution of all data points assigned to each expert. Whereas the original ME formulation used expectations of assignment variables called *responsibilities*, this is inadequate for inference in the mixture of GP experts. Consequently, we directly represent the indicators, $c_i$, and Gibbs sample for them to capture their dependencies.

In Gibbs sampling we need the posterior conditional distribution for each indicator given all the remaining indicators and the data:

$$p(c_i = j|\mathbf{c}_{-i},\mathbf{x},\mathbf{y},\theta,\phi) \propto p(\mathbf{y}|c_i = j, \mathbf{c}_{-i}, \mathbf{x}, \theta)p(c_i = j|\mathbf{c}_{-i},\mathbf{x},\phi),$$

where $\mathbf{c}_{-i}$ denotes all indicators except number $i$. We defer discussion of the second term defining the gating network to the next section. As discussed, the first term being the likelihood given the indicators factors into independent terms for each expert. For Gibbs sampling we therefore need the probability of output $y_i$ under GP number $j$:

$$p(y_i|\{y_\ell : \ell \neq i, c_\ell = j\}, \{x_\ell : c_\ell = j\},\theta_j).$$

For a GP model, this conditional density is the well known Gaussian [Williams & Rasmussen, 1996]:

$$p(y_i | \mathbf{y}_{-i}, \mathbf{x}, \theta) \sim \mathcal{N}(\mu, \sigma^2), \qquad \left\{ \begin{array}{l} \mu = Q(x_i, \mathbf{x})^\top Q^{-1} \mathbf{y}_{-i} \\ \sigma^2 = Q(x_i, x_i) - Q(x_i, \mathbf{x})^\top Q^{-1} Q(x_i, \mathbf{x}) \end{array} \right. \qquad (2)$$

where the covariance matrix $Q$ depends on the parameters $\theta$. Thus, for the GP expert, we compute the above conditional density by simply evaluating the GP on the data assigned to it. Although this equation looks computationally expensive, we can keep track of the inverse covariance matrices and reuse them for consecutive Gibbs updates by performing rank one updates (since Gibbs sampling changes at most one indicator at a time).

We are free to choose any valid covariance function for the experts. In our simulations we employed the following Gaussian covariance function:

$$Q(x_i, x_{i'}) = v_0 \exp\left(-\frac{1}{2} \sum_d (x_{id} - x_{i'd})^2 / w_d^2\right) + v_1 \delta(i, i') \qquad (3)$$

with hyperparameters $v_0$ controlling the signal variance, $v_1$ controlling the noise variance, and $w_d$ controlling the length scale or (inverse) *relevance* of the $d$-th dimension of $x$ in relation to predicting $y$; $\delta$ is the Kronecker delta function (i.e. $\delta(i, i') = 1$ if $i = i'$, o.w. 0).

## 3 The Gating network

The gating network assigns probability to different experts based entirely on the input. We will derive a gating network based on the Dirichlet Process which can be defined as the limit of a Dirichlet distribution when the number of classes tends to infinity. The standard Dirichlet Process is not input dependent, but we will modify it to serve as a gating mechanism. We start from a symmetric Dirichlet distribution on proportions:

$$p(\pi_1, ... \pi_k | \alpha) \sim \text{Dirichlet}(\alpha/k) = \frac{\Gamma(\alpha)}{\Gamma(\alpha/k)^k} \prod_j \pi_j^{\alpha/k-1},$$

where $\alpha$ is the (positive) concentration parameter. It can be shown [Rasmussen, 2000] that the conditional probability of a single indicator when integrating over the $\pi_j$ variables and letting $k$ tend to infinity is given by:

$$
\begin{array}{lll}
\text{components where } n_{-i,j} > 0: & p(c_i = j | \mathbf{c}_{-i}, \alpha) & = & \dfrac{n_{-i,j}}{n - 1 + \alpha}, \\[2ex]
\text{all other components combined:} & p(c_i \neq c_{i'} \text{ for all } i' \neq i | \mathbf{c}_{-i}, \alpha) & = & \dfrac{\alpha}{n - 1 + \alpha},
\end{array}
\qquad (4)
$$

where $n_{-i,j} (= \sum_{i' \neq i} \delta(c_{i'}, j))$ is the *occupation number* of expert $j$ excluding observation $i$, and $n$ is the total number of data points. This shows that the probabilities are proportional to the occupation numbers. To make the gating network input dependent, we will simply employ a local estimate [1] for this occupation number using a kernel classifier:

$$n_{-i,j} = (n - 1) \frac{\sum_{i' \neq i} K_\phi(x_i, x_{i'}) \delta(c_{i'}, j)}{\sum_{i' \neq i} K_\phi(x_i, x_{i'})}, \qquad (5)$$

where the delta function selects data points assigned to class $j$, and $K$ is the kernel function parametrized by $\phi$. As an example we use a Gaussian kernel function:

$$K_\phi(x_i, x_{i'}) = \exp\left(-\frac{1}{2} \sum_d (x_{id} - x_{i'd})^2 / \phi_d^2\right), \qquad (6)$$

parameterized by length scales $\phi_d$ for each dimension. These length scales allow dimensions of $x$ space to be more or less relevant to the gating network classification.

We Gibbs sample from the indicator variables by multiplying the input-dependent Dirichlet process prior eq. (4) and (5) with the GP conditional density eq. (2). Gibbs sampling in an infinite model requires that the indicator variables can take on values that no other indicator variable has already taken, thereby creating new experts. We use the auxiliary variable approach of Neal [1998] (algorithm 8 in that paper). In this approach hyperparameters for new experts are sampled from their prior and the likelihood is evaluated based on these. This requires finding the likelihood of a Gaussian process with no data. Fortunately, for the covariance function eq. (3) this likelihood is Gaussian with zero mean and variance $v_0 + v_1$.

If all $n$ data points are assigned to a single GP, the likelihood calculation will still be cubic in the number of data points (per Gibbs sweep over all indicators). We can reduce the computational complexity by introducing the constraint that no GP expert can have more than $n_{max}$ data points assigned to it. This is easily implemented[2] by modifying the conditionals in the Gibbs sampler.

The hyperparameter $\alpha$ controls the prior probability of assigning a data point to a new expert, and therefore influences the total number of experts used to model the data. As in Rasmussen [2000], we give a vague inverse gamma prior to $\alpha$, and sample from its posterior using Adaptive Rejection Sampling (ARS) [Gilks & Wild, 1992]. Allowing $\alpha$ to vary gives the model more freedom to infer the number of GPs to use for a particular dataset.

Finally we need to do inference for the parameters of the gating function. Given a set of indicator variables one could use standard methods from kernel classification to optimize the kernel widths in different directions. These methods typically optimize the leave-one-out pseudo-likelihood (ie the product of the conditionals), since computing the likelihood in a model defined purely from conditional distributions as in eq. (4), (5) & (6) is generally difficult (and as pointed out in the discussion section there may not even be a single likelihood). In our model we multiply the pseudo-likelihood by a (vague) prior and sample from the resulting pseudo-posterior.

## 4   The Algorithm

The individual GP experts are given a stationary Gaussian covariance function, with a single length scale per dimension, a signal variance and a noise variance, i.e. $D + 2$ (where $D$ is the dimension of the input) hyperparameters per expert, eq. (3). The signal and noise variances are given inverse gamma priors with hyper-hypers $a$ and $b$ (separately for the two variances). This serves to couple the hyperparameters between experts, and allows the priors on $v_0$ and $v_1$ (which are used when evaluating auxiliary classes) to adapt. Finally we give vague independent log normal priors to the lenght scale paramters $w$ and $\phi$.

The algorithm for learning an infinite mixture of GP experts consists of the following steps:

1. Initialize indicator variables $c_i$ to a single value (or a few values if individual GPs are to be kept small for computational reasons).
2. Do a Gibbs sampling sweep over all indicators.
3. Do Hybrid Monte Carlo (HMC) [Duane et al, 1987] for hyperparameters of the GP covariance function, $v_0, v_1, w_d$, for each expert in turn. We used 10 leapfrog iterations with a stepsize small enough that rejections were rare.
4. Optimize the hyper-hypers, $a$ & $b$, for each of the variance parameters.
5. Sample the Dirichlet process concentration parameter, $\alpha$ using ARS.

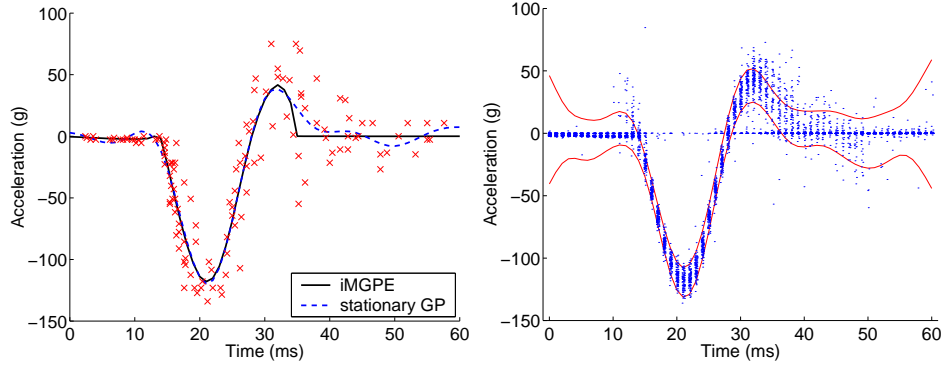

Figure 1: The left hand plot shows the motorcycle impact data (133 points) together with the median of the model's predictive distribution, and for comparison the mean of a stationary covariance GP model (with optimized hyperparameters). On the right hand plot we show 100 samples from the posterior distribution for the iMGPE of the (noise free) function evaluated intervals of 1 ms. We have jittered the points in the plot along the time dimension by adding uniform $\pm 0.2$ ms noise, so that the density can be seen more easily. Also, the $\pm 2$ std error (95%) confidence interval for the (noise free) function predicted by a stationary GP is plotted (thin lines).

6. Sample the gating kernel widths, $\phi$; we use the Metropolis method to sample from the pseudo-posterior with a Gaussian proposal fit at the current $\phi$[3]

7. Repeat from 2 until the Markov chain has adequately sampled the posterior.

## 5  Simulations on a simple real-world data set

To illustrate our algorithm, we used the motorcycle dataset, fig. 1, discussed in Silverman [1985]. This dataset is obviously non-stationary and has input-dependent noise. We noticed that the raw data is discretized into bins of size $\epsilon \simeq 1.2$ g; accordingly we cut off the prior for the noise variance at $\epsilon^2/12$.

The model is able to capture the general shape of the function and also the input-dependent nature of the noise (fig. 1). This can be seen from the right hand plot in fig. 1, where the uncertainty of the function is very low for $t < 10$ ms owing to a small inferred noise level in this region. For comparison, the predictions from a stationary GP has been superimposed in fig. 1. Whereas the medians of the predictive distributions agree to a large extent (left hand plot), we see a huge difference in the predictive distributions (right hand). The homoscedastic GP cannot capture the very tight distribution for $t < 10$ ms offered by iMGPE. Also for large $t > 40$ ms, the iMGPE model predicts with fairly high probability that the signal could be very close to zero. Note that the predictive distribution of the function is multimodal, for example, around time 35 ms. Multimodal predictive distributions could in principle be obtained from an ordinary GP by integrating over hyperparameters, however, in a mixture of GP's model they can arise naturally. The predictive distribution of the function appears to have significant mass around 0 g which seems somewhat artifactual. We explicitly did not normalize or center the data, which has a large range in output. The

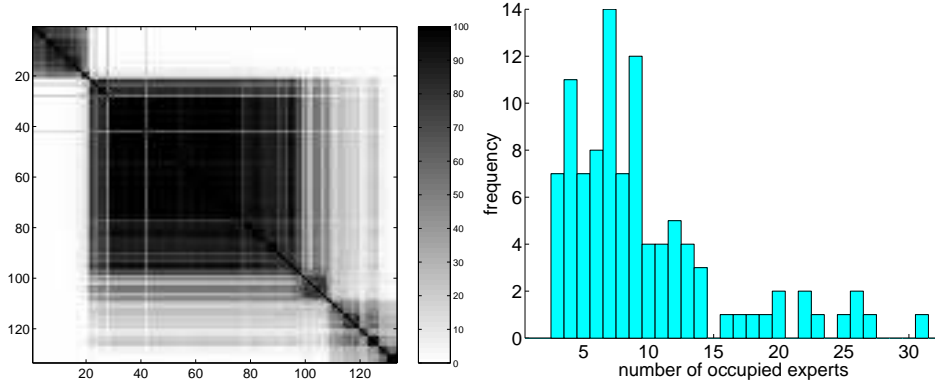

Figure 2: The left hand plot shows the number of times, out of 100 samples, that the indicator variables for two data points were equal. The data have been sorted from left-to-right according to the value of the time variable (since the data is not equally spaced in time the axis of this matrix cannot be aligned with the plot in fig.1). The right hand plot shows a histogram over the 100 samples of the number of GP experts used to model the data.

Gaussian processes had zero mean a priori, which coupled with the concentration of data around zero may explain the posterior mass at zero. It would be more natural to treat the GP means as separate hyperparameters controlled by a hyper-hyperparameter (centered at zero) and do inference on them, rather than fix them all at 0. Although the scatter of data from the predictive distribution for iMGPE looks somewhat messy with multimodality etc, it is important to note that it assigns high density to regions that seem probable.

The motorcycle data appears to have roughly three regions: a flat low-noise region, followed by a curved region, and a flat high noise region. This intuition is bourne out by the model. We can see this in two ways. Fig 2. (left) shows the number of times two data points were assigned to the same expert. A clearly defined block captures the initial flat region and a few other points that lie near the 0 g line; the middle block captures the curved region, with a more gradual transition to the last flat region. A histogram of the number of GP experts used shows that the posterior distribution of number of needed GPs has a broad peak between 3 and 10, where less than 3 occupied experts is very unlikely, and above 10 becoming progressively less likely. Note that it never uses just a single GP to model the data which accords with the intuition that a single stationary covariance function would be inadequate. We should point out that the model is not trying to do model selection between finite GP mixtures, but rather always assumes that there are infinitely many available, most of which contribute with small mass[4] to a diffuse density in the background.

In figure 3 we assessed the convergence rate of the Markov Chain by plotting the auto-correlation function for several parameters. We conclude that the mixing time is around 100 iterations[5]. Consequently, we run the chain for a total of 11000 iterations, discarding the initial 1000 (burn-in) and keeping every 100'th after that. The total computation time was around 1 hour (1 GHz Pentium).

The right hand panel of figure 3 shows the distribution of the gating function kernel width

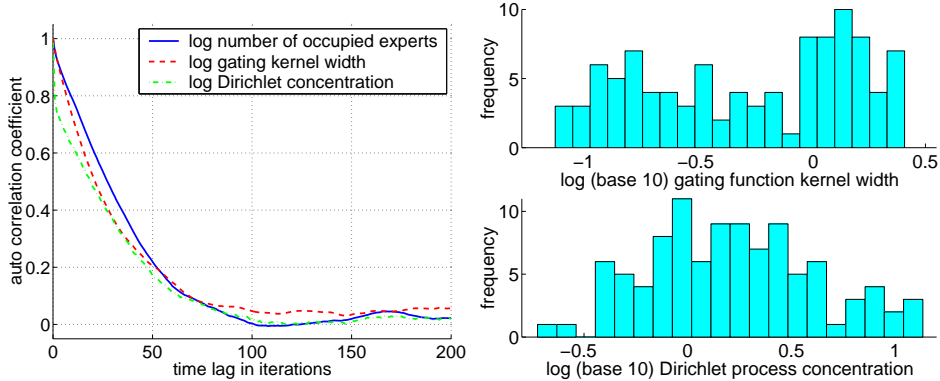

Figure 3: The left hand plot shows the auto-correlation for various parameters of the model based on 11000 iterations. The right hand plots show the distribution of the (log) kernel width $\phi$ and (log) Dirichlet concentration parameter $\alpha$, based on 100 samples from the posterior.

$\phi$ and the concentration parameter of the Dirichlet process. The posterior[6] kernel width $\phi$ lies between 0.1 and 3; comparing to the scale of the inputs these are quite short distances, corresponding to rapid transitions between experts (as opposed to lengthy intervals with multiple active experts). This corresponds well with our visual impression of the data.

## 6    Discussion and Conclusions

We now return to Tresp [2000]. There are four ways in which the infinite mixture of GP experts differs from, and we believe, improves upon the model presented by Tresp. First, in his model, although a gating network divides up the input space, each GP expert predicts on the basis of *all* of the data. Data that was not assigned to a GP expert can therefore spill over into predictions of a GP, which will lead to bias near region boundaries especially for experts with long length scales. Second, if there are $M$ experts, Tresp's model has $3M$ GPs (the experts, noise models, and separate gating functions) each of which requires computations over the entire dataset resulting in $O(3Mn^3)$ computations. In our model since the experts divide up the data points, if there are $M$ experts equally dividing the data an iteration takes $O(n^3/M)$ computations (each of $n$ Gibbs updates requires a rank-one computation $O(n^2/M^2)$ for each of the $M$ experts and the Hybrid Monte Carlo takes $M$ times $O(n^3/M^3)$). Even for modest $M$ (e.g. $M \approx 10$) this can be a significant saving. Inference for the gating length scale parameters is $O(n^2 D^2)$ if the full Hessian is used, but can be reduced to $O(n^2 D)$ for a diagonal approximation, or Hybrid Monte Carlo if the input dimension is large. Third, by going to the Dirichlet process infinite limit, we allow the model to infer the number of components required to capture the data. Finally, in our model the GP hyperparameters are not fixed but are instead inferred from the data.

We have defined the gating network prior implicitly in terms of the conditional distribution of an indicator variable given all the other indicator variables. Specifically, the distribution of this indicator variable is an input-dependent Dirichlet process with counts given by local estimates of the data density in each class eq. (5). We have not been able to prove that these conditional distributions are always consistent with a single joint distribution over

the indicators. If indeed there does not exist a single consistent joint distribution the Gibbs sampler may converge to different distributions depending on the order of sampling.

We are encouraged by the preliminary results obtained on the motorcycle data. Future work should also include empirical comparisons with other state-of-the-art regression methods on multidimensional benchmark datasets. We have argued here that single iterations of the MCMC inference are computationally tractable even for large data sets, experiments will show whether mixing is sufficiently rapid to allow practical application. We hope that the extra flexibility of the effective covariance function will turn out to improve performance. Also, the automatic choice of the number of experts may make the model advantageous for practical modeling tasks.

Finally, we wish to come back to the modeling philosophy which underlies this paper. The computational problem in doing inference and prediction using Gaussian Processes arises out of the unrealistic assumption that a single covariance function captures the behavior of the data over its entire range. This leads to a cumbersome matrix inversion over the entire data set. Instead we find that by making a more realistic assumption, that the data can be modeled by an infinite mixture of local Gaussian processes, the computational problem also decomposes into smaller matrix inversions.

## Footnotes

[1]this local estimate won't generally be an integer, but this doesn't have any adverse consequences

[2]We simply set the conditional probability of joining a class which has been deemed full to zero.

[3]The Gaussian fit uses the derivative and Hessian of the log posterior wrt the log length scales. Since this is an asymmetric proposal the acceptance probabilities must be modified accordingly. This scheme has the advantage of containing no tunable parameters; however when the dimension $D$ is large, it may be computationally more efficient to use HMC, to avoid calculation of the Hessian.

[4]The total mass of the non-represented experts is $\alpha/(n + \alpha)$, where the posterior for $\alpha$ in this experiment is peaked between 1 and 2 (see figure 3, bottom right panel), corresponding to about 1% of the total mass

[5]the sum of the auto-correlation coefficients from $-\infty$ to $\infty$ is an estimate of the mixing time

[6]for comparison the (vague) *prior* on the kernel width is log normal with 95% of the mass between 0.1 and 100, corresponding to very short (sub sample) distances upto distances comparable to the entire input range

### References

Gibbs, M. N. (1997). Bayesian Gaussian Processes for Regression and Classification. PhD thesis. University of Cambridge.

Goldberg, P. W., Williams, C. K. I., & Bishop C. M. (1998). Regression with Input-dependent Noise, NIPS 10.

Duane, S., Kennedy, A. D., Pendleton, B. J., and Roweth, D. (1987). Hybrid Monte Carlo, *Physics letters B*, vol. 55, pp. 2774–2777.

Gilks, W. R. & Wild, P. (1992). Adaptive rejection sampling for Gibbs sampling. *Applied Statistics 41*, 337–348.

Jacobs, R. A., Jordan, M. I., Nowlan, S. J. & Hinton, G. E. (1991). Adaptive mixture of local experts. *Neural Computation*, vol 3, pp 79–87.

Neal, R. M. (1998). Markov chain sampling methods for Dirichlet process mixture models. Technical Report 4915, Department of Statistics, University of Toronto. http://www.cs.toronto.edu/~radford/mixmc.abstract.html.

Rasmussen, C. E. (2000). The Infinite Gaussian Mixture Model, NIPS 12, S.A. Solla, T.K. Leen and K.-R. Müller (eds.), pp. 554–560, MIT Press.

Silverman, B. W. (1985). Some aspects of the spline smoothing approach to non-parametric regression curve fitting. *J. Royal Stat. Society. Ser. B*, vol. 47, pp. 1–52.

Smola A. J. and Bartlett, P. (2001). Sparse Greedy Gaussian Process Regression, NIPS 13.

Tresp V. (2001). Mixtures of Gaussian Process, NIPS 13.

Williams, C. K. I. and Seeger, M. (2001). Using the Nyström Method to Speed Up Kernel Machines, NIPS 13.

Williams, C. K. I. and C. E. Rasmussen (1996). Gaussian Processes for Regression, in D. S. Touretzky, M. C. Mozer and M. E. Hasselmo (editors), NIPS 8, MIT Press.
